# An Analog VLSI Neural Network for Phase-based Machine Vision

**Bertram E. Shi**
Department of Electrical and Electronic
Engineering
Hong Kong University of Science and
Technology
Clear Water Bay, Kowloon, Hong Kong

**Kwok Fai Hui**
Fujitsu Microelectronics Pacific Asia Ltd.
Suite 1015-20, Tower 1
Grand Century Place
193 Prince Edward Road West
Mongkok, Kowloon, Hong Kong.

## Abstract

We describe the design, fabrication and test results of an analog CMOS VLSI neural network prototype chip intended for phase-based machine vision algorithms. The chip implements an image filtering operation similar to Gabor-filtering. Because a Gabor filter's output is complex valued, it can be used to define a phase at every pixel in an image. This phase can be used in robust algorithms for disparity estimation and binocular stereo vergence control in stereo vision and for image motion analysis. The chip reported here takes an input image and generates two outputs at every pixel corresponding to the real and imaginary parts of the output.

## 1  INTRODUCTION

Gabor filters are used as preprocessing stages for different tasks in machine vision and image processing. Their use has been partially motivated by findings that two dimensional Gabor filters can be used to model receptive fields of orientation selective neurons in the visual cortex (Daugman, 1980) and three dimensional spatio-temporal Gabor filters can be used to model biological image motion analysis (Adelson, 1985).

A Gabor filter has a complex valued impulse response which is a complex exponential modulated by a Gaussian function. In one dimension,

$$g(x) = \frac{1}{\sqrt{2\pi}\sigma}e^{-\frac{x^2}{2\sigma^2}}e^{j\omega_{xo}x} = \frac{1}{\sqrt{2\pi}\sigma}e^{-\frac{x^2}{2\sigma^2}}(\cos(\omega_{xo}x)+j\sin(\omega_{xo}x))$$

where $\omega_{xo}$ and $\sigma$ are real constants corresponding to the angular frequency of the complex exponential and the standard deviation of the Gaussian.

The phase of the complex valued filter output at a given pixel is related to the location of edges and other features in the input image near that pixel. Because translating the image input results in a phase shift in the Gabor output, several authors have developed "phase-based" approaches to disparity estimation (Westelius, 1995) and binocular vergence control (Theimer, 1994) in stereo vision and image motion analysis (Fleet, 1992). Barron et. al.'s comparison (Barron, 1992) of algorithms for optical flow estimation indicates that Fleet's algorithm is the most accurate among those tested.

The remainder of this paper describes the design, fabrication and test results of a prototype analog VLSI continuous time neural network which implements a complex valued filter similar to the Gabor.

## 2 NETWORK AND CIRCUIT ARCHITECTURE

The prototype implements a Cellular Neural Network (CNN) architecture for Gabor-type image filtering (Shi, 1996). It consists of an array of neurons, called "cells," each corresponding to one pixel in the image to be processed. Each cell has two outputs $v_r(n)$ and $v_i(n)$ which evolve over time according to the equation

$$\begin{bmatrix} \dot{v}_r(n) \\ \dot{v}_i(n) \end{bmatrix} = \begin{bmatrix} \cos\omega_{xo} & -\sin\omega_{xo} \\ \sin\omega_{xo} & \cos\omega_{xo} \end{bmatrix}\begin{bmatrix} v_r(n-1) \\ v_i(n-1) \end{bmatrix} - \begin{bmatrix} 2+\lambda^2 & 0 \\ 0 & 2+\lambda^2 \end{bmatrix}\begin{bmatrix} v_r(n) \\ v_i(n) \end{bmatrix} + \begin{bmatrix} \cos\omega_{xo} & \sin\omega_{xo} \\ -\sin\omega_{xo} & \cos\omega_{xo} \end{bmatrix}\begin{bmatrix} v_r(n+1) \\ v_i(n+1) \end{bmatrix} + \begin{bmatrix} \lambda^2 u(n) \\ 0 \end{bmatrix}$$

where $\lambda > 0$ and $\omega_o \in [0, 2\pi]$ are real constants and $u(n)$ is the input image. The feedback from neighbouring cells' outputs enables information to be spread globally throughout the array. This network has a unique equilibrium point where the outputs correspond to the real and imaginary parts of the result of filtering the image with a complex valued discrete space convolution kernel which can be approximated by

$$g(n) = \frac{\lambda}{2} e^{-\lambda|n|} e^{j\omega_{xo}(n)}.$$

The Gaussian function of the Gabor filter has been replaced by $(\lambda/2) e^{-\lambda|x|}$. The larger $\lambda$ is, the narrower the impulse response and the larger the bandwidth. Figure 1 shows the real (a) and imaginary (b) parts of $g(n)$ for $\lambda = 0.3$ and $\omega_{xo} = 0.93$. The dotted lines show the function which modulates the complex exponential.

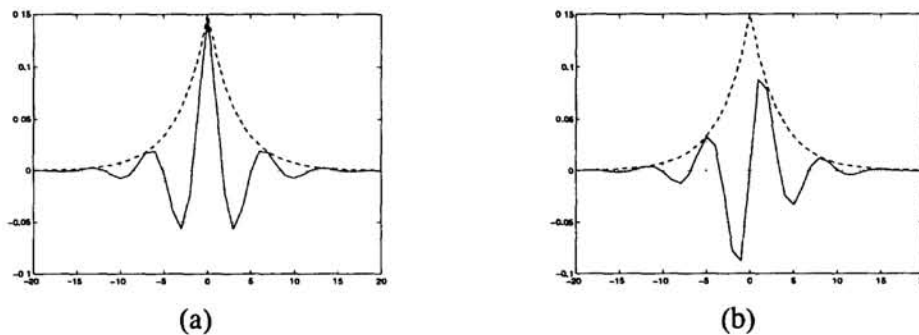

(a)  (b)

Figure 1: The Real and Imaginary Parts of the Impulse Response.

In the circuit implementation of this CNN, each output corresponds to the voltage across a capacitor. We selected the circuit architecture in Figure 2 because it was the least sensitive to the effects of random parameter variations among those we considered (Hui, 1996). In the figure, resistor labels denote conductances and trapezoidal blocks represent transconductance amplifiers labelled by their gains.

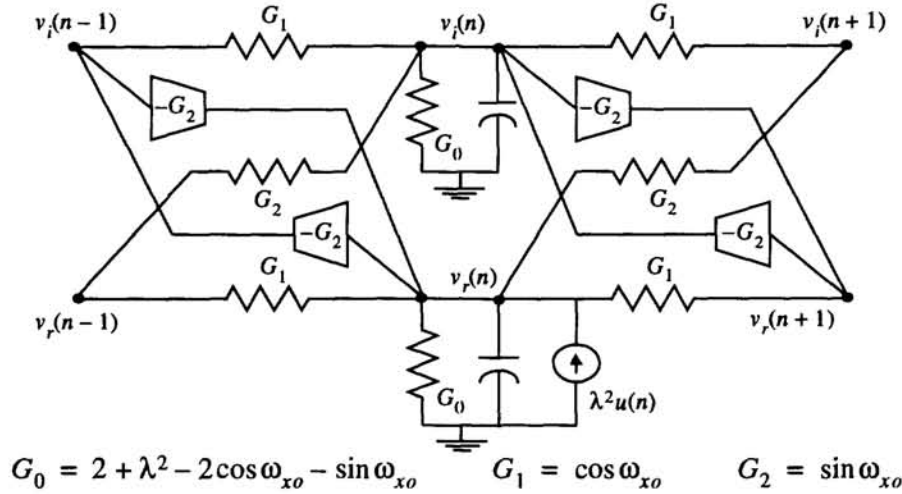

$$G_0 = 2 + \lambda^2 - 2\cos\omega_{xo} - \sin\omega_{xo} \qquad G_1 = \cos\omega_{xo} \qquad G_2 = \sin\omega_{xo}$$

Figure 2: Circuit Implementation of One Neuron.

The circuit implementation also gives good intuitive understanding of the CNN's operation. Assume that the input image is an impulse at pixel $n$. In the circuit, this corresponds to setting the current source $\lambda^2 u(n)$ to $\lambda^2$ amps and setting the remaining current sources to zero. If the gains and conductances were chosen so that $\lambda = 0.3$ and $\omega_{xo} = 0.93$, then the steady state voltages across the lower capacitors would follow the spatial distribution shown in Figure 1(a) where the center peak occurs at cell $n$ and the voltages across the upper capacitors would follow the distribution shown in Figure 1(b). To see how this would arise in the circuit, consider the current supplied by the source $\lambda^2 u(n)$. Part of the current flows through the conductance $G_0$ pushing the voltage $v_r(n)$ positive. As this voltage increases, the two resistors with conductance $G_1$ cause a smoothing effect which pulls the voltages $v_r(n-1)$ and $v_r(n+1)$ up towards $v_r(n)$. Current also flows through the diagonal resistor with conductance $G_2$ pulling $v_i(n+1)$ positive as well. At the same time, the transconductance amplifier with input $v_r(n)$ draws current from node $v_i(n-1)$ pushing $v_i(n-1)$ negative. The larger $G_2$, the more the voltages at nodes $v_i(n-1)$ and $v_i(n+1)$ are pushed negative and positive. On the other hand, the larger $G_1$, the greater the smoothing between nodes. Thus, the larger the ratio

$$\frac{G_2}{G_1} = \frac{\sin\omega_{xo}}{\cos\omega_{xo}} = \tan\omega_{xo},$$

the higher the spatial frequency $\omega_{xo}$ at which the impulse response oscillates.

## 3   DESIGN OF CMOS BUILDING BLOCKS

This section describes CMOS transistor circuits which implement the transconductance amplifiers and resistors in Figure 2. It is not necessary to implement the capacitors explicitly. Since the equilibrium point of the CNN is unique, the parasitic capacitances of the circuit are sufficient to ensure the circuit operates correctly.

### 3.1   TRANSCONDUCTANCE AMPLIFIER

The transconductance amplifiers can be implemented using the circuit shown in Figure 3(a). For $V_{in} \approx V_{GND}$, the output current is approximately $I_{out} = \sqrt{\beta_n I_{SS}} V_{in}$ where $\beta_n = \mu_n C_{ox}(W/L)$ and $(W/L)$ is the width/length ratio of the differential pair. The transistors in the current mirrors are assumed to be matched. Using cascoded current mir-

rors decreases static errors such as offsets caused by the finite output impedance of the MOS transistors in saturation.

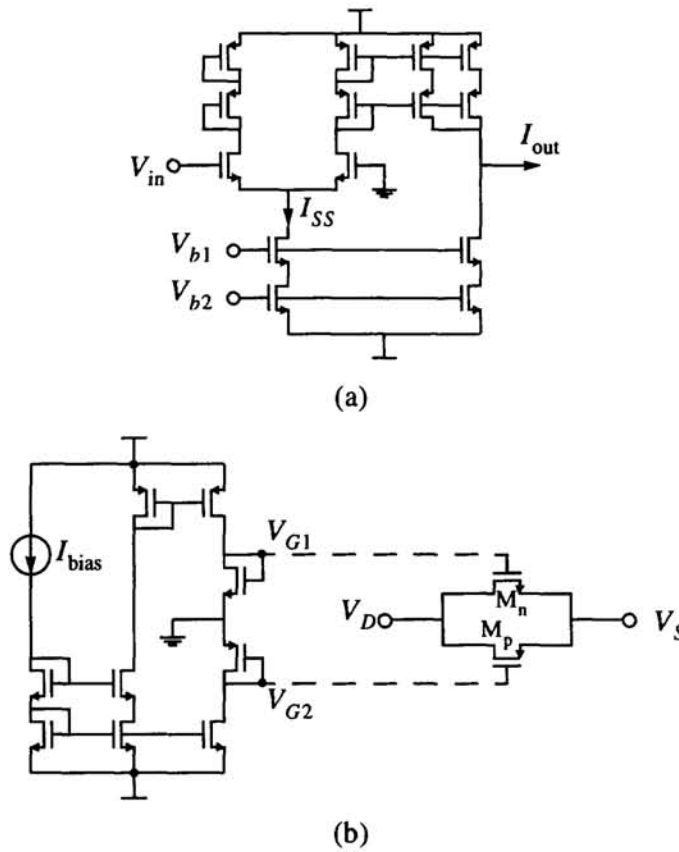

(a)

(b)

Figure 3: The CMOS Circuits Implementing OTAs and Resistors

## 3.2 RESISTORS

Since the convolution kernels implemented are modulated sine and cosine functions, the nodal voltages $v_e(n)$ and $v_o(n)$ can be both positive and negative with respect to the ground potential. The resistors in the circuit must be floating and exhibit good linearity and invariance to common mode offsets for voltages around the ground potential. Many MOS resistor circuits require bias circuitry implemented at every resistor. Since for image processing tasks, we are interested in maximizing the number of pixels processed, eliminating the need for bias circuitry at each cell will decrease its area and in turn increase the number of cells implementable within a given area.

Figure 3(b) shows a resistor circuit which satisfies the requirements above. This circuit is essentially a CMOS transmission gate with adjustable gate voltages. The global bias circuit which generates the gate voltages in the CMOS resistor is shown on the left. The gate bias voltages $V_{G1}$ and $V_{G2}$ are distributed to each resistor designed with the same value. Both transistors $M_n$ and $M_p$ operate in the conduction region where (Enz, 1995)

$$I_{Dn} = n_n\beta_n\left(V_{Pn} - \frac{V_D + V_S}{2}\right)(V_D - V_S) \text{ and } I_{Dp} = -n_p\beta_p\left(V_{Pp} - \frac{V_D + V_S}{2}\right)(V_D - V_S)$$

and $V_{Pn}$ and $V_{Pp}$ are nonlinear functions of the gate and threshold voltages. The sizing of the NMOS and PMOS transistors can be chosen to decrease the effect of the nonlinearity

due to the $(V_D + V_S)/2$ terms. The conductance of the resistors can be adjusted using $I_{bias}$.

## 3.3   LIMITATIONS

Due to the physical constraints of the circuit realizations, not all values of $\lambda$ and $\omega_{xo}$ can be realized. Because the conductance values are non-negative and the OTA gains are non-positive both $G_1$ and $G_2$ must be non-negative. This implies that $\omega_{xo}$ must lie between 0 and $\pi/2$. Because the conductance $G_0$ is non-negative, $\lambda^2 \geq -2 + 2\cos\omega_{xo} + \sin\omega_{xo}$. Figure 4 shows the range of center frequencies $\omega_{xo}$ (normalized by $\pi$) and relative bandwidths $(2\lambda/\omega_{xo})$ achievable by this realization. Not all bandwidths are achievable for $\omega_{xo} \leq 2\mathrm{atan}\,0.5 \approx 0.3\pi$.

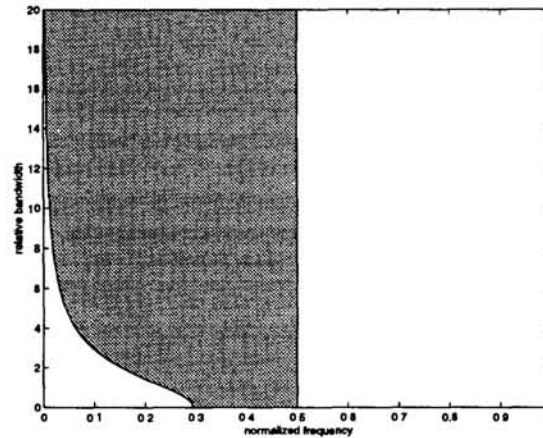

Figure 4:  The filter parameters implementable by the circuit realization.

## 4   TEST RESULTS

The circuit architecture and CMOS building blocks described above were fabricated using the Orbit 2$\mu$m n-well process available through MOSIS. In this prototype, a 13 cell one dimensional array was fabricated on a 2.2mm square die. The value of $\omega_{xo}$ is fixed at $2\mathrm{atan}\,0.5 \approx 0.927$ by transistor sizing. This is the smallest spatial frequency for which all bandwidths can be obtained. In addition, $G_0 = \lambda^2$ for this value of $\omega_{xo}$. The width of the impulse response is adjustable by changing the externally supplied bias current shown in Figure 3(b) controlling $G_0$.

The transconductance amplifiers and resistors are designed to operate between $\pm 300$mV. The currents representing the input image are provided by transconductance amplifiers internal to the chip which are controlled by externally applied voltages. Outputs are read off the chip in analog form through two common read-out amplifiers: one for the real part of the impulse response and one for the imaginary part. The outputs of the cells are connected in turn to the inputs of the read-out amplifier through transmission gates controlled by a shift register. The chip requires $\pm 4$V supplies and dissipates 35mW.

To measure the impulse response of the filters, we applied 150mV to the input corresponding to the middle cell of the array and 0V to the remaining inputs. The output voltages from one chip as a function of cell number are shown as solid lines in Figure 5(a, b). To correct for DC offsets, we also measured the output voltages when all of the inputs were grounded, as shown by the dashed lines in the figure. The DC offsets can be separated into two components: a constant offset common to all cells in the array and a small offset which varies from cell to cell. For the chip shown, the constant offset is approximately

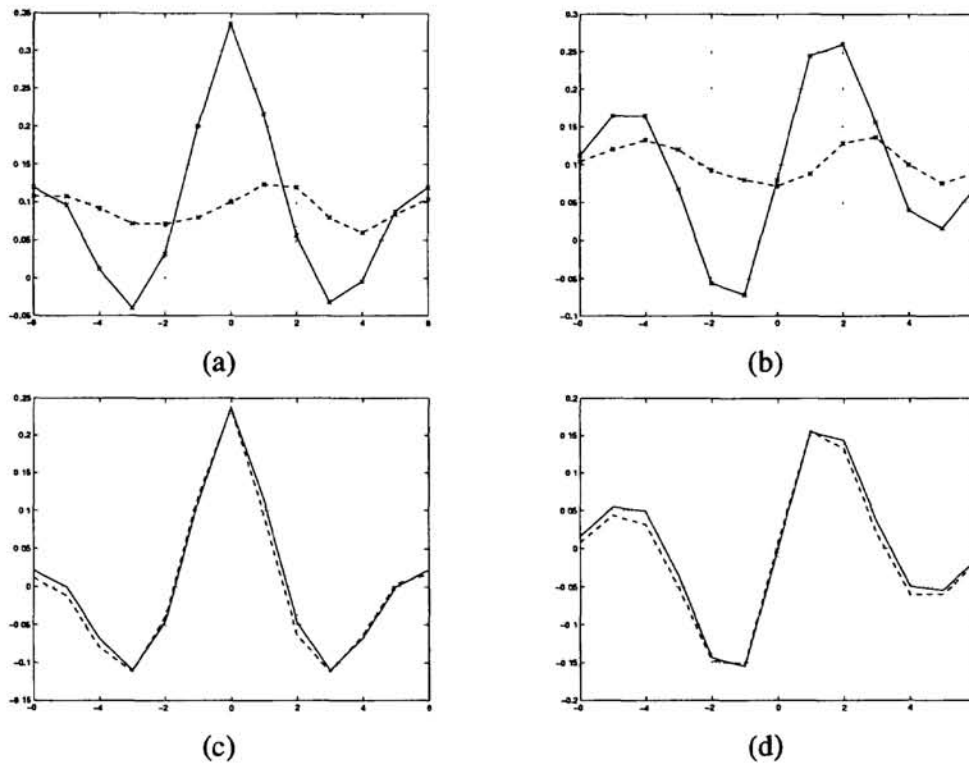

Figure 5:  DC Measurements from the Prototype

100mV and the small variations have a standard deviation of 20mV. These results are consistent with the other chips. The constant offset is primarily due to the offset voltage in the read-out amplifier. The small variations from cell to cell are the result of both parameter variations from cell to cell and offsets in the transconductance amplifiers of each cell.

By subtracting the DC zero-input offsets at each cell from the outputs, we can observe that the impulse response closely matches that predicted by the theory. The dotted lines in Figure 5(c, d) show the offset corrected outputs for the same chip as shown in Figure 5(a, b). The solid lines shows the theoretical output of the chip using parameters $\lambda$ and $\omega_{x_0}$ chosen to minimize the mean squared error between the theory and the data. The chip was designed for $\lambda = 0.210$ and $\omega_{x_0} = 0.927$. The parameters for the best fit are $\lambda = 0.175$ and $\omega_{x_0} = 0.941$. The signal to noise ratio, as defined by the energy in the theoretical output divided by the energy in the error between theory and data, is 19.3dB. Similar measurements from two other chips gave signal to noise ratios of 29.0dB ($\lambda = 0.265$, $\omega_{xo} = 0.928$) and 30.6dB ($\lambda = 0.200$, $\omega_{xo} = 0.938$).

To measure the speed of the chips, we grounded all of the inputs except that of the middle cell to which we attached a function generator generating a square wave switching between $\pm 200$mV. The rise times (10% to 90%) at the output of the chip for each cell were measured and ranged between 340 and 528 nanoseconds. The settling times will not increase if the number of cells increases since the outputs are computed in parallel. The settling time is primarily determined by the width of the impulse response. The wider the impulse response, the farther information must propagate through the array and the slower the settling time.

# 5    CONCLUSION

We have described the architecture, design and test results from an analog VLSI prototype of a neural network which filters images with convolution kernels similar to those of the Gabor filter. Our future work on chip design includes fabricating chips with larger numbers of cells, two dimensional arrays and chips with integrated photosensors which acquire and process images simultaneously. We are also investigating the use of these neural network chips in binocular vergence control of an active stereo vision system.

## Acknowledgements

This work was supported by the Hong Kong Research Grants Council (RGC) under grant number HKUST675/95E.

## References

E. H. Adelson, and J. R. Bergen, "Spatiotemporal energy models for the perception of motion", J. *Optical Society of America A*, vol. 2, pp. 284-299, Feb. 1985.

J. Barron, D. S. Fleet, S. S. Beauchemin, and T. A. Burkitt, "Performance of optical flow techniques," in *Proc. of CVPR*, (Champaign, IL), pp. 236-242, IEEE, 1992.

J. G. Daugman, "Two-dimensional spectral analysis of cortical receptive field profiles," *Vision Research*, vol. 20, pp. 847-856, 1980.

C. C. Enz, F. Krummenacher, and E. A. Vittoz, "An analytical MOS transistor model valid in all regions of operation and dedicated to low-voltage and low-current applications," *Analog Integrated Circuits and Signal Processing,* vol.8, no.1, p83-114, Jul. 1995.

D. J. Fleet, *Measurement of Image Velocity*, Boston. MA: Kluwer Academic Publishers, 1992.

K. F. Hui and B. E. Shi, "Robustness of CNN Implementations for Gabor-type Filtering," *Proc. of Asia Pacific Conference on Circuits and Systems*, pp. 105-108, Nov. 1996.

B. E. Shi, "Gabor-type image filtering with cellular neural networks," *Proceedings of the 1996 IEEE International Symposium on Circuits and Systems*, vol. 3, pp. 558-561, May 1996.

W. M. Theimer and H. A Mallot, "Phase-based binocular vergence control and depth reconstruction using active vision," *CVGIP: Image Understanding*, vol. 60, no. 3, pp. 343-358, Nov. 1994.

C.-J. Westelius, H. Knutsson, J. Wiklund and C.-F. Westin, "Phase-based disparity estimation," in J. L. Crowley and H. I. Christensen, eds., *Vision as Process*, chap. 11, pp. 157-178, Springer-Verlag, Berlin, 1995.